# Learning in Feedforward Networks with Nonsmooth Functions

**Nicholas J. Redding***
Information Technology Division
Defence Science and Tech. Org.
P.O. Box 1600 Salisbury
Adelaide SA 5108 Australia

**T. Downs**
Intelligent Machines Laboratory
Dept of Electrical Engineering
University of Queensland
Brisbane Q 4072 Australia

## Abstract

This paper is concerned with the problem of learning in networks where some or all of the functions involved are not smooth. Examples of such networks are those whose neural transfer functions are piecewise-linear and those whose error function is defined in terms of the $\ell_\infty$ norm.

Up to now, networks whose neural transfer functions are piecewise-linear have received very little consideration in the literature, but the possibility of using an error function defined in terms of the $\ell_\infty$ norm has received some attention. In this latter work, however, the problems that can occur when gradient methods are used for nonsmooth error functions have not been addressed.

In this paper we draw upon some recent results from the field of nonsmooth optimization (NSO) to present an algorithm for the nonsmooth case. Our motivation for this work arose out of the fact that we have been able to show that, in backpropagation, an error function based upon the $\ell_\infty$ norm overcomes the difficulties which can occur when using the $\ell_2$ norm.

## 1 INTRODUCTION

This paper is concerned with the problem of learning in networks where some or all of the functions involved are not smooth. Examples of such networks are those whose neural transfer functions are piecewise-linear and those whose error function is defined in terms of the $\ell_\infty$ norm.

Up to now, networks whose neural transfer functions are piecewise-linear have received very little consideration in the literature, but the possibility of using an error function defined in terms of the $\ell_\infty$ norm has received some attention [1]. In the work described in [1], however, the problems that can occur when gradient methods are used for nonsmooth error functions have not been addressed.

In this paper we draw upon some recent results from the field of nonsmooth optimization (NSO) to present an algorithm for the nonsmooth case. Our motivation for this work arose out of the fact that we have been able to show [2][1] that an error function based upon the $\ell_\infty$ norm overcomes the difficulties which can occur when using backpropagation's $\ell_2$ norm [4].

The framework for NSO is the class of locally Lipschitzian functions [5]. Locally Lipschitzian functions are a broad class of functions that include, but are not limited to, "smooth" (completely differentiable) functions. (Note, however, that this framework does not include step-functions.) We here present a method for training feedforward networks (FFNs) whose behaviour can be described by a locally Lipschitzian function $\mathbf{y} = \mathbf{f}_{\text{net}}(\mathbf{w}, \mathbf{x})$, where the input vector $\mathbf{x} = (x_1, \ldots, x_n)$ is an element of the set of patterns $\mathcal{X} \subset \mathbf{R}^n$, $\mathbf{w} \in \mathbf{R}^b$ is the weight vector, and $\mathbf{y} \in \mathbf{R}^m$ is the $m$-dimensional output.

The possible networks that fit within the locally Lipschitzian framework include any network that has a continuous, piecewise differentiable description, *i.e.*, continuous functions with nondifferentiable points ("nonsmooth functions").

Training a network involves the selection of a weight vector $\mathbf{w}^*$ which minimizes an *error function* $E(\mathbf{w})$. As long as the error function $E$ is locally Lipschitzian, then it can be trained by the procedure that we will outline, which is based upon a new technique for NSO [6].

In Section 2, a description of the difficulties that can occur when gradient methods are applied to nonsmooth problems is presented. In Section 3, a short overview of the Bundle-Trust algorithm [6] for NSO is presented. And in Section 4 details of applying a NSO procedure to training networks with an $\ell_\infty$ based error function are presented, along with simulation results that demonstrate the viability of the technique.

## 2   FAILURE OF GRADIENT METHODS

Two difficulties which arise when gradient methods are applied to nonsmooth problems will be discussed here. The first is that gradient descent sometimes fails to converge to a local minimum, and the second relates to the lack of a stopping criterion for gradient methods.

### 2.1   THE "JAMMING" EFFECT

We will now show that gradient methods can fail to converge to a local minimum (the "jamming" effect [7,8]). The particular example used here is taken from [9].

Consider the following function, that has a minimum at the point $\mathbf{w}^* = (0,0)$:

$$f_1(\mathbf{w}) = 3(w_1^2 + 2w_2^2). \qquad (1)$$

If we start at the point $\mathbf{w}_0 = (2,1)$, it is easily shown that a steepest descent algorithm[2] would generate the sequence $\mathbf{w}_1 = (2,-1)/3$, $\mathbf{w}_2 = (2,1)/9, \ldots$, so that the sequence

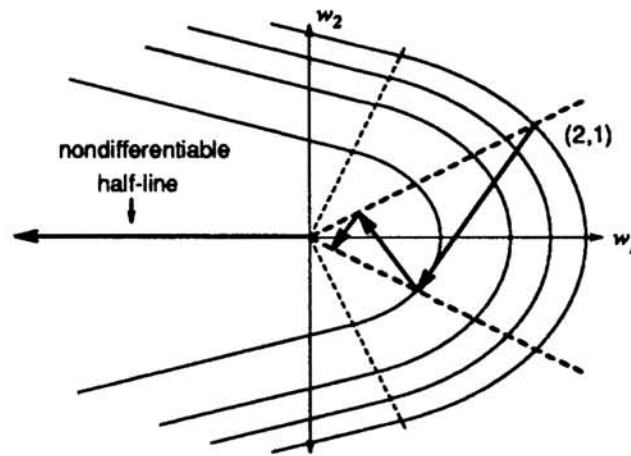

Figure 1: A contour plot of the function $f_3$.

$\{\mathbf{w}_k\}$ oscillates between points on the two half-lines $w_1 = 2w_2$ and $w_1 = -2w_2$ for $w_1 \geqslant 0$, converging to the optimal point $\mathbf{w}^* = (0,0)$. Next, from the function $f_1$, create a new function $f_2$ in the following manner:

$$f_2(\mathbf{w}) = \sqrt{f_1} = \sqrt{3(w_1^2 + 2w_2^2)}. \tag{2}$$

The gradient at any point of $f_2$ is proportional to the gradient at the same point on $f_1$, so the sequence of points generated by a gradient descent algorithm starting from $(2,1)$ on $f_2$ will be the same as the case for $f_1$, and will again converge[3] to the optimal point, again $\mathbf{w}^* = (0,0)$.

Lastly, we shift the optimal point away from $(0,0)$, but keep a region including the sequence $\{\mathbf{w}_k\}$ unchanged to create a new function $f_3(\mathbf{w})$:

$$f_3(\mathbf{w}) = \begin{cases} \sqrt{3(w_1^2 + 2w_2^2)} & \text{if } 0 \leqslant |w_2| \leqslant 2w_1 \\ \frac{1}{\sqrt{3}}(w_1 + 4|w_2|) & \text{elsewhere.} \end{cases} \tag{3}$$

The new function $f_3$, depicted in fig. 1, is continuous, has a discontinuous derivative only on the half-line $w_1 \leqslant 0$, $w_2 = 0$, and is convex with a "minimum" as $w_1 \to -\infty$. In spite of this, the steepest descent algorithm still converges to the now nonoptimal "jamming" point $(0,0)$. A multitude of possible variations to $f_1$ exist that will achieve a similar result, but the point is clear: gradient methods can lead to trouble when applied to nonsmooth problems.

This lesson is important, because the backpropagation learning algorithm is a smooth gradient descent technique, and as such will have the difficulties described when it, or an extension (*eg.*, [1]), are applied to a nonsmooth problem.

## 2.2  STOPPING CRITERION

The second significant problem associated with smooth descent techniques in a nonsmooth context occurs with the stopping criterion. In normal smooth circumstances, a stopping

criterion is determined using

$$\|\nabla f\| \leqslant \epsilon, \tag{4}$$

where $\epsilon$ is a small positive quantity determined by the required accuracy. However, it is frequently the case that the minimum of a nonsmooth function occurs at a nondifferentiable point or "kink", and the gradient is of little value around these points. For example, the gradient of $f(w) = |w|$ has a magnitude of 1 no matter how close $w$ is to the optimum at $w = 0$.

# 3    NONSMOOTH OPTIMIZATION

For any locally Lipschitzian function $f$, the *generalized directional derivative* always exists, and can be used to define a *generalized gradient* or *subdifferential*, denoted by $\partial f$, which is a compact convex set[4] [5]. A particular element $\mathbf{g} \in \partial f(\mathbf{w})$ is termed a *subgradient* of $f$ at $\mathbf{w}$ [5,10]. In situations where $f$ is strictly differentiable at $\mathbf{w}$, the generalized gradient of $f$ at $\mathbf{w}$ is equal to the gradient, *i.e.*, $\partial f(\mathbf{w}) = \nabla f(\mathbf{w})$.

We will now discuss the basic aspects of NSO and in particular the Bundle-Trust (BT) algorithm [6].

Quite naturally, subgradients in NSO provide a substitute for the gradients in standard smooth optimization using gradient descent. Accordingly, in an NSO procedure, we require the following to be satisfied:

$$\text{At every } \mathbf{w}, \text{ we can compute } f(\mathbf{w}) \text{ and any } \mathbf{g} \in \partial f(\mathbf{w}). \tag{5}$$

To overcome the jamming effect, however, it is not sufficient replace the gradient with a subgradient in a gradient descent algorithm — the strictly local information that this provides about the function's behaviour can be misleading. For example, an approach like this will not change the descent path taken from the starting point $(2, 1)$ on the function $f_3$ (see fig. 1).

The solution to this problem is to provide some "smearing" of the gradient information by enriching the information at $\mathbf{w}$ with knowledge of its surroundings. This can be achieved by replacing the strictly local subgradients $\mathbf{g} \in \partial f(\mathbf{w})$ by $\bigcup_{\mathbf{v} \in B} \mathbf{g} \in \partial f(\mathbf{v})$ where $B$ is a suitable neighbourhood of $\mathbf{w}$, and then define the *$\epsilon$-generalized gradient* $\partial_\epsilon f(\mathbf{w})$ as

$$\partial_\epsilon f(\mathbf{w}) \overset{\triangle}{=} \text{co} \left\{ \bigcup_{\mathbf{v} \in B(\mathbf{w}, \epsilon)} \partial f(\mathbf{v}) \right\} \tag{6}$$

where $\epsilon > 0$ and small, and co denotes a convex hull. These ideas were first used by [7] to overcome the lack of continuity in minimax problems, and have become the basis for extensive work in NSO.

In an optimization procedure, points in a sequence $\{\mathbf{w}_k, k = 0, 1, \ldots\}$ are visited until a point is reached at which a stopping criterion is satisfied. In a NSO procedure, this occurs when a point $\mathbf{w}_k$ is reached that satisfies the condition $0 \in \partial_\epsilon f(\mathbf{w}_k)$, and the point is said to be *$\epsilon$-optimal*. That is, in the case of convex $f$, the point $\mathbf{w}_k$ is *$\epsilon$-optimal* if

$$f(\mathbf{w}_k) \leqslant f(\mathbf{w}) + \epsilon\|\mathbf{w} - \mathbf{w}_k\| + \epsilon \text{ for all } \mathbf{w} \in \Re^n \tag{7}$$

and in the case of nonconvex $f$,

$$f(\mathbf{w}_k) \leqslant f(\mathbf{w}) + \epsilon\|\mathbf{w} - \mathbf{w}_k\| + \epsilon \text{ for all } \mathbf{w} \in B \tag{8}$$

where $B$ is some neighbourhood of $\mathbf{w}_k$ of nonzero dimension. Obviously, as $\epsilon \to 0$, then $\mathbf{w}_k \to \mathbf{w}^*$ at which $0 \in \partial f(\mathbf{w}^*)$, *i.e.*, $\mathbf{w}_k$ is "within $\epsilon$" of the local minimum $\mathbf{w}^*$.

Usually the $\epsilon$-generalized gradient is not available, and this is why the *bundle concept* is introduced. The basic idea of a bundle concept in NSO is to replace the $\epsilon$-generalized gradient by some inner approximating polytope $P$ which will then be used to compute a descent direction. If the polytope $P$ is a sufficiently good approximation to $f$, then we will find a direction along which to descend (a so-called *serious step*). In the case where $P$ is not a sufficiently good approximation to $f$ to yield a descent direction, then we perform a *null step*, staying at our current position $\mathbf{w}$, and try to improve $P$ by adding another subgradient $\partial f(\mathbf{v})$ at some nearby point $\mathbf{v}$ to our current position $\mathbf{w}$.

A natural way of approximating $f$ is by using a *cutting plane* (CP) *approximation*. The CP approximation of $f(\mathbf{w})$ at the point $\mathbf{w}_k$ is given by the expression [6]

$$\max_{1 \leqslant i \leqslant k} \{\mathbf{g}_i^t(\mathbf{w} - \mathbf{w}_i) + f(\mathbf{w}_i)\}, \tag{9}$$

where $\mathbf{g}_i$ is a subgradient of $f$ at the point $\mathbf{w}_i$. We see then that (9) provides a piecewise linear approximation of convex[5] $f$ from below, which will coincide with $f$ at all points $\mathbf{w}_i$. For convenience, we redefine the CP approximation in terms of $\mathbf{d} = \mathbf{w} - \mathbf{w}_k$, $\mathbf{d} \in \mathbf{R}^b$, the vector difference of the point of approximation, $\mathbf{w}$, and the current point in the optimization sequence, $\mathbf{w}_k$, giving the CP approximation $f_{CP}$ of $f$:

$$f_{CP}(\mathbf{w}_k, \mathbf{d}) = \max_{1 \leqslant i \leqslant k} \{\mathbf{g}_i^t\mathbf{d} + \mathbf{g}_i^t(\mathbf{w}_k - \mathbf{w}_i) + f(\mathbf{w}_i)\}. \tag{10}$$

Now, when the CP approximation is minimized to find a descent direction, there is no reason to trust the approximation far away from $\mathbf{w}_k$. So, to discourage a large step size, a stabilizing term $\frac{1}{2t_k}\mathbf{d}^t\mathbf{d}$, where $t_k$ is positive, is added to the CP approximation.

If the CP approximation at $\mathbf{w}_k$ of $f$ is good enough, then the $\mathbf{d}_k$ given by

$$\mathbf{d}_k = \arg\min_{\mathbf{d}} f_{CP}(\mathbf{w}_k, \mathbf{d}) + \frac{1}{2t_k}\mathbf{d}^t\mathbf{d} \tag{11}$$

will produce a descent direction such that a line search along $\mathbf{w}_k + \lambda\mathbf{d}_k$ will find a new point $\mathbf{w}_{k+1}$ at which $f(\mathbf{w}_{k+1}) < f(\mathbf{w}_k)$ (a *serious step*). It may happen that $f_{CP}$ is such a poor approximation of $f$ that a line search along $\mathbf{d}_k$ is not a descent direction, or yields only a marginal improvement in $f$. If this occurs, a *null step* is taken and one enriches the *bundle* of subgradients from which the CP approximation is computed by adding a subgradient from $\partial f(\mathbf{w}_k + \lambda\mathbf{d}_k)$ for small $\lambda > 0$. Each *serious step* guarantees a decrease in $f$, and a stopping criterion is provided by terminating the algorithm as soon as $\mathbf{d}_k$ in (11) satisfies the $\epsilon$-optimality criterion, at which point $\mathbf{w}_k$ is $\epsilon$-optimal. These details are the basis of bundle methods in NSO [9,10].

The bundle method described suffers from a weak point: its success depends on the delicate selection of the parameter $t_k$ in (11) [6]. This weakness has led to the incorporation of a "trust region" concept [11] into the bundle method to obtain the BT (bundle-trust) algorithm [6].

To incorporate a trust region, we define a "radius" that defines a ball in which we can "trust" that $f_{CP}$ is a good approximation of $f$. In the BT algorithm, by following trust region concepts, the choice of $t_k$ is not made *a priori* and is determined during the algorithm by varying $t_k$ in a systematic way (*trust* part) and improving the CP approximation by *null steps* (*bundle* part) until a satisfactory CP approximation $f_{CP}$ is obtained along with a ball (in terms of $t_k$) on which we can *trust* the approximation. Then the $d_k$ in (11) will lead to a substantial decrease in $f$.

The full details of the BT algorithm can be found in [6], along with convergence proofs.

# 4   EXAMPLES

## 4.1   A SMOOTH NETWORK WITH NONSMOOTH ERROR FUNCTION

The particular network example we consider here is a two-layer FFN (*i.e.*, one with a single layer of hidden units) where each output unit's value $y_i$ is computed from its discriminant function $Q_{o_i} = w_{i0} + \sum_{j=1}^h w_{ij} z_j$, by the transfer function $y_i = \tanh(Q_{o_i})$, where $z_j$ is the output of the $j$-th hidden unit. The $j$-th hidden unit's output $z_j$ is given by $z_j = \tanh(Q_{h_j})$, where $Q_{h_j} = v_{j0} + \sum_{k=1}^n v_{jk} x_j$ is its discriminant function. The $\ell_\infty$ error function (which is locally Lipschitzian) is defined to be

$$E(\mathbf{w}) = \max_{\mathbf{x} \in \mathcal{X}} \max_{1 \leqslant i \leqslant m} |Q_{o_i}(\mathbf{x}) - t_i(\mathbf{x})|, \tag{12}$$

where $t_i(\mathbf{x})$ is the desired output of output unit $i$ for the input pattern $\mathbf{x} \in \mathcal{X}$.

To make use of the BT algorithm described in the previous section, it is necessary to obtain an expression from which a subgradient at $\mathbf{w}$ for $E(\mathbf{w})$ in (12) can be computed. Using the generalized gradient calculus in [5, Proposition 2.3.12], a subgradient $g \in \partial E(\mathbf{w})$ is given by the expression[6]

$$g = \text{sgn}\left(Q_{o_{i'}}(\mathbf{x}') - t_{i'}(\mathbf{x}')\right) \nabla_{\mathbf{w}} Q_{o_{i'}}(\mathbf{x}') \quad \text{for some } i', \mathbf{x}' \in \mathcal{J} \tag{14}$$

where $\mathcal{J}$ is the set of patterns and output indices for which $E(\mathbf{w})$ in (12) obtains it maximum value, and the gradient $\nabla_{\mathbf{w}} Q_{o_{i'}}(\mathbf{x}')$ is given by

$$\nabla_{\mathbf{w}} Q_{o_{i'}}(\mathbf{x}') = \begin{cases} 1 & \text{w.r.t. } w_{i'0} \\ z_j & \text{w.r.t. } w_{i'j} \\ (1 - z_j^2) w_{i'j} & \text{w.r.t. } v_{j0} \\ x_k'(1 - z_j^2) w_{i'j} & \text{w.r.t. } v_{jk} \\ 0 & \text{elsewhere.} \end{cases} \tag{15}$$

(Note that here $j = 1, 2, \ldots, h$ and $k = 1, \ldots, n$).

The BT technique outlined in the previous section was applied to the standard XOR and 838 encoder problems using the $\ell_\infty$ error function in (12) and subgradients from (14,15).

$$\partial f(w) = \begin{cases} 1 & w > 0 \\ \text{co}\{1, -1\} & x = 0 \\ -1 & x < 0 \end{cases} \tag{13}$$

and a suitable subgradient $g \in \partial f(w)$ can be obtained by choosing $g = \text{sgn}(w)$.

In all test runs, the BT algorithm was run until convergence to a local minimum of the $\ell_\infty$ error function occurred with $\epsilon$ set at $10^{-4}$. On the XOR problem, over 20 test runs using a randomly initialized 2-2-1 network, an average of 52 function and subgradient evaluations were required. The minimum number of function and subgradient evaluations required in the test runs was 23 and the maximum was 126. On the 838 encoder problem, over 20 test runs using a randomly initialized 8-3-8 network, an average of 334 function and subgradient evaluations were required. For this problem, the minimum number of function and subgradient evaluations required in the test runs was 221 and the maximum was 512.

## 4.2   A NONSMOOTH NETWORK AND NONSMOOTH ERROR FUNCTION

In this section we will consider a particular example that employs a network function that is nonsmooth as well as a nonsmooth error function (the $\ell_\infty$ error function of the previous example).

Based on the piecewise-linear network employed by [12], let the $i$-th output of the network be given by the expression

$$y_i = \sum_{k=1}^{n} u_{ik} x_k + \sum_{j=1}^{h} w_{ij} \left| \sum_{k=1}^{n} v_{jk} x_k + v_{j0} \right| + w_{i0} \tag{16}$$

with an $\ell_\infty$-based error function

$$E(\mathbf{w}) = \max_{\mathbf{x} \in \mathcal{X}} \max_{1 \leqslant i \leqslant m} |y_i(\mathbf{x}) - t_i(\mathbf{x})|. \tag{17}$$

Once again using the generalized gradient calculus from [5, Proposition 2.3.12], a single subgradient $\mathbf{g} \in \partial E(\mathbf{w})$ is given by the expression

$$\mathbf{g} = \operatorname{sgn}(y_{i'}(\mathbf{x}') - t_{i'}(\mathbf{x}')) \begin{cases} x_k & \text{w.r.t. } u_{i'k} \\ 1 & \text{w.r.t. } w_{i'0} \\ \left| \sum_{k=1}^{n} v_{jk} x_k + v_{j0} \right| & \text{w.r.t. } w_{i'j} \\ w_{i'j} \operatorname{sgn}(\sum_{k=1}^{n} v_{jk} x_k + v_{j0}) & \text{w.r.t. } v_{j0} \\ w_{i'j} \operatorname{sgn}(\sum_{k=1}^{n} v_{jk} x_k + v_{j0}) x_k' & \text{w.r.t. } v_{jk} \\ 0 & \text{elsewhere.} \end{cases} \tag{18}$$

(Note that $j = 1, 2, \ldots, h, k = 1, 2, \ldots, n$).

In all cases the $\epsilon$-stopping criterion is set at $10^{-4}$. On the XOR problem, over 20 test runs using a randomly initialized 2-2-1 network, an average of 43 function and subgradient evaluations were required. The minimum number of function and subgradient evaluations required in the test runs was 30 and the maximum was 60. On the 838 encoder problem, over 20 test runs using a randomly initialized 8-3-8 network, an average of 445 function and subgradient evaluations were required. For this problem, the minimum number of function and subgradient evaluations required in the test runs was 386 and the maximum was 502.

## 5   CONCLUSIONS

We have demonstrated the viability of employing NSO for training networks in the case where standard procedures, with their implicit smoothness assumption, would have difficulties or find impossible. The particular nonsmooth examples we considered involved an error function based on the $\ell_\infty$ norm, for the case of a network with sigmoidal characteristics and a network with a piecewise-linear characteristic.

Nonsmooth optimization problems can be dealt with in many different ways. A possible alternative approach to the one presented here (that works for most NSO problems) is to express the problem as a composite function and then solve it using the exact penalty method (termed *composite NSO*) [11]. Fletcher [11, p. 358] states that in practice this can require a great deal of storage or be too complicated to formulate. In contrast, the BT algorithm solves the more general *basic NSO* problem and so can be more widely applied than techniques based on composite functions. The BT algorithm is simpler to set up, but this can be at the cost of algorithm complexity and a computational overhead. The BT algorithm, however, does retain the gradient descent flavour of backpropagation because it uses the generalized gradient concept along with a chain rule for computing these (generalized) gradients. Nongradient-based and stochastic methods for NSO do exist, but they were not considered here because they do not retain the gradient-based deterministic flavour. It would be useful to see if these other techniques are faster for practical problems.

The message should be clear however — smooth gradient techniques should be treated with suspicion when a nonsmooth problem is encountered, and in general the more complicated nonsmooth methods should be employed.

## Footnotes

*The author can be contacted via email at internet address redding@itd.dsto.oz.au.

[1]This is quite simple, using a theorem due to Krishnan [3].

[2]This is achieved by repeatedly performing a line search along the steepest descent direction.

[3]Note that for this new sequence of points, the gradient no longer converges to 0 at $(0,0)$, but oscillates between the values $\sqrt{2}(1, \pm 1)$.

[4]In other words, a set of vectors will define the generalized gradient of a nonsmooth function at a single point, rather than a single vector in the case of smooth functions.

[5]In the nonconvex $f$ case, (9) is not an approximation to $f$ from below, and additional tolerance parameters must be considered to accommodate this situation [6].

[6]Note that for a function $f(w) = |w| = \max\{w, -w\}$, the generalized gradient is given by the expression

## References

[1] P. Burrascano, "A norm selection criterion for the generalized delta rule," *IEEE Transactions on Neural Networks* 2 (1991), 125–130.

[2] N. J. Redding, "Some Aspects of Representation and Learning in Artificial Neural Networks," University of Queensland, PhD Thesis, June, 1991.

[3] T. Krishnan, "On the threshold order of a Boolean function," *IEEE Transactions on Electronic Computers* EC-15 (1966), 369–372.

[4] M. L. Brady, R. Raghavan & J. Slawny, "Backpropagation fails to separate where perceptrons succeed," *IEEE Transactions on Circuits and Systems* 36 (1989).

[5] F. H. Clarke, *Optimization and Nonsmooth Analysis*, Canadian Mathematical Society Series of Monographs and Advanced Texts, John Wiley & Sons, New York, NY, 1983.

[6] H. Schramm & J. Zowe, "A version of the bundle idea for minimizing a nonsmooth function: conceptual ideas, convergence analysis, numerical results," *SIAM Journal on Optimization* (1991), to appear.

[7] V. F. Dem'yanov & V. N. Malozemov, *Introduction to Minimax*, John Wiley & Sons, New York, NY, 1974.

[8] P. Wolfe, "A method of conjugate subgradients for minimizing nondifferentiable functions," in *Nondifferentiable Optimization*, M. L. Balinski & P. Wolfe, eds., Mathematical Programming Study #3, North-Holland, Amsterdam, 1975, 145–173.

[9] C. Lemaréchal, "Nondifferentiable Optimization," in *Optimization*, G. L. Nemhauser, A. H. G. Rinnooy Kan & M. J. Todd, eds., Handbooks in Operations Research and Management Science #1, North-Holland, Amsterdam, 1989, 529–572.

[10] K. C. Kiwiel, *Methods of Descent for Nondifferentiable Optimization*, Lect. Notes in Math. #1133, Springer-Verlag, New York–Heidelberg–Berlin, 1985.

[11] R. Fletcher, *Practical Methods of Optimization* second edition, John Wiley & Sons, New York, NY, 1987.

[12] R. Batruni, "A multilayer neural network with piecewise-linear structure and backpropagation learning," *IEEE Transactions on Neural Networks* 2 (1991), 395–403.
